# Recovering a Feed-Forward Net From Its Output

Charles Fefferman[*] and Scott Markel
David Sarnoff Research Center
CN5300
Princeton, NJ 08543-5300
e-mail: cf@math.princeton.edu
smarkel@sarnoff.com

## ABSTRACT

We study feed-forward nets with arbitrarily many layers, using the standard sigmoid, tanh x. Aside from technicalities, our theorems are: 1. Complete knowledge of the output of a neural net for arbitrary inputs uniquely specifies the architecture, weights and thresholds; and 2. There are only finitely many critical points on the error surface for a generic training problem.

Neural nets were originally introduced as highly simplified models of the nervous system. Today they are widely used in technology and studied theoretically by scientists from several disciplines. However, they remain little understood.

Mathematically, a (feed-forward) neural net consists of:

(1)     A finite sequence of positive integers $(D_0, D_1, \ldots, D_L)$;

(2)     A family of real numbers $(\omega_{jk}^{\ell})$ defined for $1 \le \ell \le L$, $1 \le j \le D_\ell$, $1 \le k \le D_{\ell-1}$; and

(3)     A family of real numbers $(\theta_j^{\ell})$ defined for $1 \le \ell \le L$, $1 \le j \le D_\ell$.

The sequence $(D_0, D_1, \ldots, D_L)$ is called the *architecture* of the neural net, while the $\omega_{jk}^{\ell}$ are called *weights* and the $\theta_j^{\ell}$ *thresholds*.

Neural nets are used to compute non-linear maps from $\mathbb{R}^N$ to $\mathbb{R}^M$ by the following construction. We begin by fixing a nonlinear function $\sigma(x)$ of one variable. Analogy with the nervous system suggests that we take $\sigma(x)$ asymptotic to constants as $x$ tends to $\pm\infty$; a standard choice, which we adopt throughout this paper, is $\sigma(x) =$

---

[*]Alternate address: Dept. of Mathematics, Princeton University, Princeton, NJ 08544-1000.

$\tanh\left(\frac{1}{2}x\right)$. Given an "input" $(t_1,\ldots,t_{D_0}) \in \mathbb{R}^{D_0}$, we define real numbers $x_j^\ell$ for $0 \leq \ell \leq L$, $1 \leq j \leq D_\ell$ by the following induction on $\ell$.

(4)    If $\ell = 0$ then $x_j^\ell = t_j$.

(5)    If the $x_k^{\ell-1}$ are known with $\ell$ fixed $(1 \leq \ell \leq L)$, then we set

$$x_j^\ell = \sigma\left(\sum_{1 \leq k \leq D_{\ell-1}} \omega_{jk}^\ell x_k^{\ell-1} + \theta_j^\ell\right) \quad \text{for} \quad 1 \leq j \leq D_\ell.$$

Here $x_1^\ell,\ldots,x_{D_\ell}^\ell$ are interpreted as the outputs of $D_\ell$ "neurons" in the $\ell^{\text{th}}$ "layer" of the net. The *output map* of the net is defined as the map

(6)    $\Phi: (t_1,\ldots,t_{D_0}) \longmapsto (x_1^L,\ldots,x_{D_L}^L)$.

In practical applications, one tries to pick the neural net $[(D_0, D_1,\ldots,D_L), (\omega_{jk}^\ell), (\theta_j^\ell)]$ so that the output map $\Phi$ approximates a given map about which we have only imperfect information. The main result of this paper is that under generic conditions, perfect knowledge of the output map $\Phi$ uniquely specifies the architecture, the weights and the thresholds of a neural net, up to obvious symmetries. More precisely, the obvious symmetries are as follows. Let $(\gamma_0, \gamma_1,\ldots,\gamma_L)$ be permutations, with $\gamma_\ell: \{1,\ldots,D_\ell\} \to \{1,\ldots,D_\ell\}$; and let $\{\varepsilon_j^\ell: 0 \leq \ell \leq L, 1 \leq j \leq D_\ell\}$ be a collection of $\pm 1$'s. Assume that $\gamma_\ell = (\text{identity})$ and $\varepsilon_j^\ell = +1$ whenever $\ell = 0$ or $\ell = L$. Then one checks easily that the neural nets

(7)    $[(D_0, D_1,\ldots,D_L), (\omega_{jk}^\ell), (\theta_j^\ell)]$    and

(8)    $[(D_0, D_1,\ldots,D_L), (\widetilde{\omega}_{jk}^\ell), (\widetilde{\theta}_j^\ell)]$

have the same output map if we set

(9)    $\widetilde{\omega}_{jk}^\ell = \varepsilon_j^\ell \omega_{[\gamma_\ell j][\gamma_{\ell-1} k]}^\ell \varepsilon_k^{\ell-1}$    and    $\widetilde{\theta}_j^\ell = \varepsilon_j^\ell \theta_{[\gamma_\ell j]}^\ell$.

This reflects the facts that the neurons in layer $\ell$ are interchangeable $(1 \leq \ell \leq L-1)$, and that the function $\sigma(x)$ is odd. The nets (7) and (8) will be called *isomorphic* if they are related by (9). Note in particular that isomorphic neural nets have the same architecture. Our main theorem asserts that, under generic conditions, any two neural nets with the same output map are isomorphic.

We discuss the generic conditions which we impose on neural nets. We have to avoid obvious counterexamples such as:

(10)    Suppose all the weights $\omega_{jk}^\ell$ are zero. Then the output map $\Phi$ is constant. The architecture and thresholds of the neural net are clearly not uniquely determined by $\Phi$.

(11)    Fix $\ell_0, j_1, j_2$ with $1 \leq \ell_0 \leq L - 1$ and $1 \leq j_1 < j_2 \leq D_{\ell_0}$. Suppose we have $\theta_{j_1}^{\ell_0} = \theta_{j_2}^{\ell_0}$ and $\omega_{j_1 k}^{\ell_0} = \omega_{j_2 k}^{\ell_0}$ for all $k$. Then (5) gives $x_{j_1}^{\ell_0} = x_{j_2}^{\ell_0}$. Therefore, the

output depends on $\omega_{jj_1}^{\ell_0+1}$ and $\omega_{jj_2}^{\ell_0+1}$ only through the sum $\omega_{jj_1}^{\ell_0+1} + \omega_{jj_2}^{\ell_0+1}$. So the output map does not uniquely determine the weights.

Our hypotheses are more than adequate to exclude these counterexamples. Specifically, we assume that

(12) $\theta_j^\ell \neq 0$ and $|\theta_j^\ell| \neq |\theta_{j'}^\ell|$ for $j \neq j'$.

(13) $\omega_{jk}^\ell \neq 0$; and for $j \neq j'$, the ratio $\omega_{jk}^\ell / \omega_{j'k}^\ell$ is not equal to any fraction of the form $p/q$ with $p$, $q$ integers and $1 \leq q \leq 100 \, D_\ell^2$.

Evidently, these conditions hold for generic neural nets. The precise statement of our main theorem is as follows. *If two neural nets satisfy* (12), (13) *and have the same output, then the nets are isomorphic.* It would be interesting to replace (12), (13) by minimal hypotheses, and to study functions $\sigma(x)$ other than $\tanh\left(\frac{1}{2}x\right)$.

We now sketch the proof of our main result, sacrificing accuracy for simplicity. After a trivial reduction, we may assume $D_0 = D_L = 1$. Thus, the outputs of the nodes $x_j^\ell(t)$ are functions of one variable, and the output map of the neural net is $t \mapsto x_1^L(t)$. The key idea is to continue the $x_j^\ell(t)$ analytically to complex values of $t$, and to read off the structure of the net from the set of singularities of the $x_j^\ell$. Note that $\sigma(x) = \tanh\left(\frac{1}{2}x\right)$ is meromorphic, with poles at the points of an arithmetic progression $\{(2m+1)\pi i : m \in \mathbb{Z}\}$. This leads to two crucial observations.

(14) When $\ell = 1$, the poles of $x_j^\ell(t)$ form an arithmetic progression $\Pi_j^1$, and

(15) When $\ell > 1$, every pole of any $x_k^{\ell-1}(t)$ is an accumulation point of poles of any $x_j^\ell(t)$.

In fact, (14) is immediate from the formula $x_j^1(t) = \sigma(\omega_{j1}^1 t + \theta_j^1)$, which is merely the special case $D_0 = 1$ of (5). We obtain

(16) $\Pi_j^1 = \left\{ \dfrac{(2m+1)\pi i - \theta_j^1}{\omega_{j1}^1} : m \in \mathbb{Z} \right\}$

To see (15), fix $\ell$, $j$, $\hat{k}$, and assume for simplicity that $x_{\hat{k}}^{\ell-1}(t)$ has a simple pole at $t_0$, while $x_k^{\ell-1}(t)$ ($k \neq \hat{k}$) is analytic in a neighborhood of $t_0$. Then

(17) $x_{\hat{k}}^{\ell-1}(t) = \dfrac{\lambda}{t - t_0} + f(t)$, with $f$ analytic in a neighborhood of $t_0$.

From (17) and (5), we obtain

(18) $x_j^\ell(t) = \sigma(\omega_{j\hat{k}}^\ell \lambda (t - t_0)^{-1} + g(t))$, with

(19) $g(t) = \omega_{j\hat{k}}^\ell f(t) + \sum_{k \neq \hat{k}} \omega_{jk}^\ell x_k^{\ell-1}(t) + \theta_j^\ell$ analytic in a neighborhood of $t_0$.

Thus, in a neighborhood of $t_0$, the poles of $x_j^\ell(t)$ are the solutions $\tilde{t}_m$ of the equation

(20) $\dfrac{\omega_{j\hat{k}}^\ell \lambda}{\tilde{t}_m - t_0} + g(\tilde{t}_m) = (2m+1)\pi i, \quad m \in \mathbb{Z}.$

There are infinitely many solutions of (20), accumulating at $t_0$. Hence. $t_0$ is an accumulation point of poles of $x_j^\ell(t)$, which completes the proof of (15).

In view of (14), (15), it is natural to make the following definitions. The *natural domain* of a neural net is the largest open subset of the complex plane to which the output map $t \mapsto x_1^L(t)$ can be analytically continued. For $\ell \geq 0$ we define the $\ell^{\text{th}}$ *singular set* $\mathrm{Sing}(\ell)$ by setting

$$\mathrm{Sing}(0) \quad = \text{ complement of the natural domain in } \mathbb{C}, \quad \text{and}$$

$$\mathrm{Sing}(\ell + 1) = \text{ the set of all accumulation points of } \mathrm{Sing}(\ell).$$

These definitions are made entirely in terms of the output map, without reference to the structure of the given neural net. On the other hand, the sets $\mathrm{Sing}(\ell)$ contain nearly complete information on the architecture, weights and thresholds of the net.

This will allow us to read off the structure of a neural net from the analytic continuation of its output map. To see how the sets $\mathrm{Sing}(\ell)$ reflect the structure of the net, we reason as follows.

From (14) and (15) we expect that

(21)   For $1 \leq \ell \leq L$, $\mathrm{Sing}(L - \ell)$ is the union over $j = 1, \ldots, D_\ell$ of the set of poles of $x_j^\ell(t)$, together with their accumulation points (which we ignore here), and

(22)   For $\ell \geq L$, $\mathrm{Sing}(\ell)$ is empty.

Immediately, then, we can read off the "depth" $L$ of the neural net; it is simply the smallest $\ell$ for which $\mathrm{Sing}(\ell)$ is empty.

We need to solve for $D_\ell$, $\omega_{jk}^\ell$, $\theta_j^\ell$. We proceed by induction on $\ell$.

*When $\ell = 1$,* (14) and (21) show that $\mathrm{Sing}(L - 1)$ is the union of arithmetic progressions $\Pi_j^1$, $j = 1, \ldots, D_1$. Therefore, from $\mathrm{Sing}(L - 1)$ we can read off $D_1$ and the $\Pi_j^1$. (We will return to this point later in the introduction.) In view of (16), $\Pi_j^1$ determines the weights and thresholds at layer 1, modulo signs. Thus. we have found $D_1$, $\omega_{jk}^1$, $\theta_j^1$.

*When $\ell > 1$,* we may assume that

(23)   The $D_{\ell'}$, $\omega_{jk}^{\ell'}$, $\theta_j^{\ell'}$ are already known, for $1 \leq \ell' < \ell$.

Our task is to find $D_\ell$, $\omega_{jk}^\ell$, $\theta_j^\ell$. In view of (23), we can find a pole $t_0$ of $x_{\hat{k}}^{\ell-1}(t)$ for our favorite $\hat{k}$. Assume for simplicity that $t_0$ is a simple pole of $x_{\hat{k}}^{\ell-1}(t)$, and that the $x_k^{\ell-1}(t)$ ($k \neq \hat{k}$) are analytic in a neighborhood of $t_0$. Then $x_{\hat{k}}^{\ell-1}(t)$ is given by (17) in a neighborhood of $t_0$, with $\lambda$ already known by virtue of (23). Let $U$ be a small neighborhood of $t_0$.

We will look at the image $Y$ of $U \cap \mathrm{Sing}(L - \ell)$ under the map $t \mapsto \frac{\lambda}{t - t_0}$. Since $\lambda$, $t_0$ and $\mathrm{Sing}(L - \ell)$ are already known, so is $Y$. On the other hand, we can relate $Y$ to $D_\ell$, $\omega_{jk}^\ell$, $\theta_j^\ell$ as follows. From (21) we see that $Y$ is the union over $j = 1, \ldots, D_\ell$ of

(24)   $Y_j = $ image of $U \cap \{$ Poles of $x_j^\ell(t)\}$ under $t \mapsto \frac{\lambda}{(t - t_0)}$.

For fixed $j$, the poles of $x_j^\ell(t)$ in a neighborhood of $t_0$ are the $\tilde{t}_m$ given by (20). We write

(25)    $$\frac{\omega_{j\hat{k}}^\ell \lambda}{\tilde{t}_m - t_0} = \left[\frac{\omega_{j\hat{k}}^\ell \lambda}{(\tilde{t}_m - t_0)} + g(\tilde{t}_m)\right] + \left[g(t_0) - g(\tilde{t}_m)\right].$$

Equation (20) shows that the first expression in brackets in (25) is equal to $(2m + 1)\pi i$. Also, since $\tilde{t}_m \to t_0$ as $|m| \to \infty$ and $g$ is analytic in a neighborhood of $t_0$, the second expression in brackets in (25) tends to zero. Hence,

$$\frac{\omega_{j\hat{k}}^\ell \lambda}{\tilde{t}_m - t_0} = (2m + 1)\pi i - g(t_0) + o(1) \quad \text{for large } m.$$

Comparing this with the definition (24), we see that $Y_j$ is asymptotic to the arithmetic progression

(26)    $$\Pi_j^\ell = \left\{\frac{(2m + 1)\pi i - g(t_0)}{\omega_{j\hat{k}}^\ell} : m \in \mathbb{Z}\right\}.$$

Thus, the known set $Y$ is the union over $j = 1.\ldots, D_\ell$ of sets $Y_j$, with $Y_j$ asymptotic to the arithmetic progression $\Pi_j^\ell$. From $Y$, we can therefore read off $D_\ell$ and the $\Pi_j^\ell$. (We will return to this point in a moment.) We see at once from (26) that $\omega_{j\hat{k}}^\ell$ is determined up to sign by $\Pi_j^\ell$. Thus, we have found $D_\ell$ and $\omega_{j\hat{k}}^\ell$. With more work, we can also find the $\theta_j^\ell$, completing the induction on $\ell$.

The above induction shows that the structure of a neural net may be read off from the analytic continuation of its output map. We believe that the analytic continuation of the output map will lead to further consequences in the study of neural nets.

Let us touch briefly on a few points which we glossed over above. First of all, suppose we are given a set $Y \subset \mathbb{C}$, and we know that $Y$ is the union of sets $Y_1, \ldots, Y_D$, with $Y_j$ asymptotic to an arithmetic progression $\Pi_j$. We assumed above that $\Pi_1, \ldots, \Pi_D$ are uniquely determined by $Y$. In fact, without some further hypothesis on the $\Pi_j$, this need not be true. For instance, we cannot distinguish $\Pi_1 \cup \Pi_2$ from $\Pi_3$ if $\Pi_1 = \{\text{odd integers}\}$, $\Pi_2 = \{\text{even integers}\}$. $\Pi_3 = \{\text{all integers}\}$. On the other hand, we can clearly recognize $\Pi_1 = \{\text{all integers}\}$ and $\Pi_2 = \{m\sqrt{2} : m \text{ an integer}\}$ from their union $\Pi_1 \cup \Pi_2$. Thus, irrational numbers enter the picture. The rôle of our generic hypothesis (13) is to control the arithmetic progressions that arise in our proof.

Secondly, suppose $x_{\hat{k}}^\ell(t)$ has a pole at $t_0$. We assumed for simplicity that $x_k^\ell(t)$ is analytic in a neighborhood of $t_0$ for $k \neq \hat{k}$. However, one of the $x_k^\ell(t)$ ($k \neq \hat{k}$) may also have a pole at $t_0$. In that case, the $x_j^{\ell+1}(t)$ may all be analytic in a neighborhood of $t_0$, because the contributions of the singularities of the $x_k^\ell$ in $\sigma\left(\sum_k \omega_{jk}^{\ell+1} x_k^\ell + \theta_j^{\ell+1}\right)$ may cancel. Thus, the singularity at $t_0$ may disappear from the output map. While this circumstance is hardly generic, it is not ruled out by our hypotheses (12), (13).

Because singularities can disappear, we have to make technical changes in our description of $\text{Sing}(\ell)$. For example, in the discussion following (23), $Y$ need not be the union of the sets $Y_j$. Rather, $Y$ is their "approximate union". (See [F]).

Next, we should point out that the signs of the weights and thresholds require some attention, even though we have some freedom to change signs by applying isomorphisms. (See (9).)

Finally, in the definition of the natural domain, we have assumed that there is a unique maximal open set to which the output map continues analytically. This need not be true of a general real-analytic function on the line – for instance. take $f(t) = (1 + t^2)^{1/2}$. Fortunately, the natural domain is well-defined for any function that continues analytically to the complement of a countable set. The defining formula (5) lets us check easily that the output map continues to the complement of a countable set, so the natural domain makes sense. This concludes our overview of the proof of our main theorem. The full proof of our results will appear in [F].

Both the uniqueness problem and the use of analytic continuation have already appeared in the neural net literature. In particular, it was R. Hecht-Nielson who pointed out the rôle of isomorphisms and posed the uniqueness problem. His paper with Chen and Lu [CLH] on "equioutput transformations" on the space of all neural nets influenced our work. E. Sontag [So] and H. Sussman [Su] proved sharp uniqueness theorems for one hidden layer. The proof in [So] uses complex variables.

## Acknowledgements

Fefferman is grateful to R. Crane, S. Markel, J. Pearson, E. Sontag, R. Sverdlove, and N. Winarsky for introducing him to the study of neural nets.

This research was supported by the Advanced Research Projects Agency of the Department of Defense and was monitored by the Air Force Office of Scientific Research under Contract F49620-92-C-0072. The United States Government is authorized to reproduce and distribute reprints for governmental purposes notwithstanding any copyright notation hereon. This work was also supported by the National Science Foundation.

The following posters, presented at NIPS 93, may clarify our uniqueness theorem.

## References

[CLH] R. Hecht-Nielson, et al., *On the geometry of feedforward neural network error surfaces.* (to appear).

[F] C. Fefferman, *Reconstructing a neural network from its output*, Revista Mathemática Iberoamericana. (to appear).

[So] F. Albertini and E. Sontag, *Uniqueness of weights for neural networks.* (to appear).

[Su] H. Sussman, *Uniqueness of the weights for minimal feedforward nets with a given input-output map*, Neural Networks **5** (1992), pp. 589–593.

## Recovering a Feed-Forward Net from Its Output

Charles Fefferman

David Sarnoff Research Center and Princeton University
Princeton, New Jersey

Presented by

Scott A. Markel
David Sarnoff Research Center
Princeton, New Jersey

e-mail addresses: cf@math.princeton.edu
smarkel@sarnoff.com

---

**Take-Home Message**

Suppose an unknown neural network is placed in a black box.

You aren't allowed to look in the box, but you are allowed to observe the outputs produced by the network for arbitrary inputs.

Then, in principle, you have enough information to determine the network architecture (number of layers and number of nodes in each layer) and the unique values for all the weights.

---

**The Output Map of a Neural Network**

Fix a feed-forward neural network with the standard sigmoid $\sigma(x) = \tanh x$.

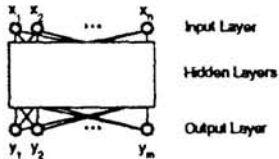

The map that carries input vectors $(x_1, ..., x_n)$

to output vectors $(y_1, ...., y_m)$

is called the OUTPUT MAP of the neural network.

---

**The Key Question**

When can two neural networks

have the same output map?

---

**Obvious Examples of Two Neural Networks with the Same Output Map**

Start with a neural network N.

Then either

1. permute the nodes in a hidden layer, or

2. fix a hidden node, and change the sign of every weight (including the bias weight) that involves that node

This yields a new neural network with the same output map as N.

---

**Uniqueness Theorem**

Let N and N' be neural networks that satisfy generic conditions described below.

If N and N' have the same output map, then they differ only by sign changes and permutations of hidden nodes.

### Generic Conditions

We assume that

- all weights are non-zero

- bias weights within each layer have distinct absolute values

- the ratio of weights from node i in layer l to nodes j and k in layer (l+1) is not equal to any fraction of the form p/q with p, q integers and 1≤q≤100*(number of nodes in layer l)

Some such assumptions are needed to avoid obvious counterexamples.

---

### Outline of the Proof

- it's enough to consider networks with one input node and one output node (see below)

- all node outputs are now functions of a single, real variable t (the network input)

- analytically continue the network output to a function f of a single, complex variable t

- the qualitative geometry of the poles of the function f determines the network architecture (see below)

- the asymptotics of the function f near its singularities determine the weights

---

### Reduction to a Network with Single Input and Output Nodes

- focus attention on a single output node, ignoring the others

- study only input data with a single non-zero entry

---

### Geometric Description of the Poles

- poles (small dots) accumulate at essential singularities (small squares)

- essential singularities (small squares) accumulate at more complicated essential singularities (large dots)

---

### Determining the Network Architecture from the Picture

- three kinds of singularities (small dots, small squares, large dots)
    ⟹ three layers of sigmoids, i.e. two hidden layers and an output layer

- three "spiral arms" of small squares accumulate at each large dot
    ⟹ three nodes in the second hidden layer

- two "spiral arms" of small dots accumulate at each small square
    ⟹ two nodes in the first hidden layer

---

### Determining the Network Architecture from the Picture (cont'd)

- from the network reduction we know that there is one input node and one output node

- therefore, the network architecture is as pictured

